# Efficient Match Kernels between Sets of Features for Visual Recognition

**Liefeng Bo**
Toyota Technological Institute at Chicago
blf0218@tti-c.org

**Cristian Sminchisescu**
University of Bonn
sminchisescu.ins.uni-bonn.de

## Abstract

In visual recognition, the images are frequently modeled as unordered collections of local features (bags). We show that bag-of-words representations commonly used in conjunction with linear classifiers can be viewed as special match kernels, which count 1 if two local features fall into the same regions partitioned by visual words and 0 otherwise. Despite its simplicity, this quantization is too coarse, motivating research into the design of match kernels that more accurately measure the similarity between local features. However, it is impractical to use such kernels for large datasets due to their significant computational cost. To address this problem, we propose efficient match kernels (EMK) that map local features to a low dimensional feature space and average the resulting vectors to form a set-level feature. The local feature maps are learned so their inner products preserve, to the best possible, the values of the specified kernel function. Classifiers based on EMK are linear both in the number of images and in the number of local features. We demonstrate that EMK are extremely efficient and achieve the current state of the art in three difficult computer vision datasets: Scene-15, Caltech-101 and Caltech-256.

## 1 Introduction

Models based on local features have achieved state-of-the art results in many visual object recognition tasks. For example, an image can be described by a set of local features extracted from patches around salient interest points or regular grids, or a shape can be described by a set of local features defined at edge points. This raises the question on how should one measure the similarity between two images represented as sets of local features. The problem is non-trivial because the cardinality of the set varies with each image and the elements are unordered.

Bag of words (BOW) [27] is probably one of the most popular image representations, due to both its conceptual simplicity and its computational efficiency. BOW represents each local feature with the closest visual word and counts the occurrence frequencies in the image. The resulting histogram is used as an image descriptor for object recognition, often in conjunction with linear classifiers. The length of the histogram is given by the number of visual words, being the same for all images. Various methods for creating vocabularies exist [10], the most common being $k$-means clustering of all (or a subsample of) the local features to obtain visual words.

An even better approach to recognition is to define kernels over sets of local features. One way is to exploit closure rules. The sum match kernel of Haussler [7] is obtained by adding local kernels over all combinations of local features from two different sets. In [17], the authors modify the sum kernel by introducing an integer exponent on local kernels. Neighborhood kernels [20] integrate the spatial location of local features into a sum match kernel. Pyramid match kernels [5, 14, 13] map local features to multi-resolution histograms and compute a weighted histogram intersection. Algebraic set kernels [26] exploit tensor products to aggregate local kernels, whereas principal angle kernels

[29] measure similarities based on angles between linear subspaces spanned by local features in the two sets. Other approaches estimate a probability distribution on sets of local features, then derive their similarity using distribution-based comparison measures [12, 18, 2]. All of the above methods need to explicitly evaluate the full kernel matrix, hence they require space and time complexity that is quadratic in the number of images. This is impractical for large datasets (see §4).

In this paper we present efficient match kernels (EMK) that combine the strengths of both bag of words and set kernels. We map local features to a low dimensional feature space and construct set-level features by averaging the resulting feature vectors. This feature extraction procedure is not significantly different than BOW. Hence EMK can be used in conjunction with linear classifiers and do not require the explicit computation of a full kernel matrix—this leads to both space and time complexity that is linear in the number of images. Experiments on three image categorization tasks show that EMK are effective computational tools.

## 2  Bag of Words and Match Kernels

In supervised image classification, we are given a training set of images and their corresponding labels. The goal is to learn a classifier to label unseen images. We adopt a bag of features method, which represents an image as a set of local features. Let $\mathbf{X} = \{\mathbf{x}_1, \ldots, \mathbf{x}_p\}$ be a set of local features in an image and $\mathbf{V} = \{\mathbf{v}_1, \ldots, \mathbf{v}_D\}$ the dictionary, a set of visual words. In BOW, each local feature is quantized into a $D$ dimensional binary indicator vector $\mu(\mathbf{x}) = [\mu_1(\mathbf{x}), \ldots, \mu_D(\mathbf{x})]^\top$. $\mu_i(\mathbf{x})$ is 1 if $\mathbf{x} \in R(\mathbf{v}_i)$ and 0 otherwise, where $R(\mathbf{v}_i) = \{\mathbf{x} : \|\mathbf{x} - \mathbf{v}_i\| \leq \|\mathbf{x} - \mathbf{v}\|, \forall \mathbf{v} \in \mathbf{V}\}$. The feature vectors for one image form a normalized histogram $\overline{\mu}(\mathbf{X}) = \frac{1}{|\mathbf{X}|} \sum_{\mathbf{x} \in \mathbf{X}} \mu(\mathbf{x})$, where $|\cdot|$ is the cardinality of a set. BOW features can be used in conjunction with either a linear or a kernel classifier, albeit the latter often leads to expensive training and testing (see §4). When a linear classifier is used, the resulting kernel function is:

$$K_B(\mathbf{X}, \mathbf{Y}) = \overline{\mu}(\mathbf{X})^\top \overline{\mu}(\mathbf{Y}) = \frac{1}{|\mathbf{X}||\mathbf{Y}|} \sum_{\mathbf{x} \in \mathbf{X}} \sum_{\mathbf{y} \in \mathbf{Y}} \mu(\mathbf{x})^\top \mu(\mathbf{y}) = \frac{1}{|\mathbf{X}||\mathbf{Y}|} \sum_{\mathbf{x} \in \mathbf{X}} \sum_{\mathbf{y} \in \mathbf{Y}} \delta(\mathbf{x}, \mathbf{y}) \quad (1)$$

with

$$\delta(\mathbf{x}, \mathbf{y}) = \begin{cases} 1, & \mathbf{x}, \mathbf{y} \subset R(\mathbf{v}_i), \exists i \in \{1, \ldots, D\} \\ 0, & \text{otherwise} \end{cases} \quad (2)$$

$\delta(\mathbf{x}, \mathbf{y})$ is obviously a positive definite kernel, measuring the similarity between two local features $\mathbf{x}$ and $\mathbf{y}$: $\delta(\mathbf{x}, \mathbf{y}) = 1$ if $\mathbf{x}$ and $\mathbf{y}$ belong the same region $R(\mathbf{v}_i)$, and 0 otherwise. However, this type of quantization can be too coarse when measuring the similarity of two local features (see also fig. 1 in [21]), risking a significant decrease in classification performance. Better would be to replace $\delta(\mathbf{x}, \mathbf{y})$ with a continuous kernel function that more accurately measures the similarity between $\mathbf{x}$ and $\mathbf{y}$:

$$K_S(\mathbf{X}, \mathbf{Y}) = \frac{1}{|\mathbf{X}||\mathbf{Y}|} \sum_{\mathbf{x} \in \mathbf{X}} \sum_{\mathbf{y} \in \mathbf{Y}} k(\mathbf{x}, \mathbf{y}) \quad (3)$$

In fact, this is related to the normalized sum match kernel [7, 17]. Based on closure properties, $K_s(\mathbf{X}, \mathbf{Y})$ is a positive definite kernel, as long as the components $k(\mathbf{x}, \mathbf{y})$ are positive definite. For convenience, we refer to $k(\mathbf{x}, \mathbf{y})$ as the local kernel. A negative impact of kernelization is the high computational cost required to compute the summation match function, which takes $O(|\mathbf{X}||\mathbf{Y}|)$ for a single kernel value rather than $O(1)$, the cost of evaluating a single kernel function defined on vectors. When used in conjunction with kernel machines, it takes $O(n^2)$ and $O(n^2 m^2 d)$ to store and compute the entire kernel matrix, respectively, where $n$ is the number of images in the training set, and $m$ is the average cardinality of all sets. For image classification, $m$ can be in the thousands of units, so the computational cost rapidly becomes quartic as $n$ approaches (or increases beyond) $m$. In addition to expensive training, the match kernel function has also a fairly high testing cost: for a test input, evaluating the discriminant $f(\mathbf{X}) = \sum_{i=1}^{n} \alpha_i K_s(\mathbf{X}_i, \mathbf{X})$ takes $O(nm^2 d)$. This is, again, unacceptably slow for large $n$. For sparse kernel machines, such as SVMs, the cost can decrease to some extent, as some of the $\alpha_i$ are zero. However, this does not change the order of complexity, as the level of sparsity usually grows linearly in $n$.

**Definition 1.** *The kernel function* $k(\mathbf{x}, \mathbf{y}) = \phi(\mathbf{x})^\top \phi(\mathbf{y})$ *is called finite dimensional if the feature map* $\phi(\cdot)$ *is finite dimensional.*

|  | Sum [7] | Bhattacharyya [12] | PMK [6] | EMK-CKSVD | EMK-Fourier |
|---|---|---|---|---|---|
| Train | $O(n^2 m^2 d)$ | $O(n^2 m^3 d)$ | $O(n^2 m \log(T) d)$ | $O(nmDd + nD^2)$ | $O(nmDd)$ |
| Test | $O(nm^2 d)$ | $O(nm^3 d)$ | $O(nm \log(T) d)$ | $O(mDd + D^2)$ | $O(mDd)$ |

Table 1: Computational complexity for five types of 'set kernels'. 'Test' means the computational cost per image.'Sum' is the sum match kernel used in [7]. 'Bhattacharyya' is the Bhattacharyya kernel in [12]. PMK is the pyramid match kernel of [6], with $T$ in PMK giving the value of the maximal feature range. $d$ is the dimensionality of local features. $D$ in EMK is the dimensionality of feature maps and does not change with the training set size. Our experiments suggest that a value of $D$ in the order of thousands of units is sufficient for good accuracy. Thus, $O(nmDd)$ will dominate the computational cost for training, and $O(mDd)$ the one for testing, since $m$ is usually in the thousands, and $d$ in the hundreds of units. EMK uses linear classifiers and does not require the evaluation of the kernel matrix. The other four methods are used in conjunction with kernel classifiers, hence they all need to evaluate the entire kernel matrix. In the case of nearest neighbor classifiers, there is no training cost, but testing costs remain unchanged.

$\delta(\mathbf{x}, \mathbf{y})$ is a special type of finite dimensional kernel. With the finite dimensional kernel, the match kernel can be simplified as:

$$K_S(\mathbf{X}, \mathbf{Y}) = \overline{\phi}(\mathbf{X})^\top \overline{\phi}(\mathbf{Y}) \tag{4}$$

where $\overline{\phi}(\mathbf{X}) = \frac{1}{|\mathbf{X}|} \sum_{\mathbf{x} \in \mathbf{X}} \phi(\mathbf{x})$ is the feature map on the set of vectors. Since $\overline{\phi}(\mathbf{X})$ is finite and can be computed explicitly, we can extract feature vectors on the set $\mathbf{X}$, then apply a linear classifier on the resulting represenation. We call (4) an efficient match kernel (EMK). The feature extraction in EMK is not significantly different from the bag of words method. The training and testing costs are $O(nmDd)$ and $O(mDd)$ respectively, where $D$ is the dimensionality of the feature map $\phi(\mathbf{x})$. If the feature map $\phi(\mathbf{x})$ is low dimensional, the computational cost of EMK can be much lower than the one required to evaluate the match kernel by computing the kernel functions $k(\mathbf{x}, \mathbf{y})$. For example, the cost is $\frac{1}{n}$ lower when $D$ has the same order as $m$ (this is the case in our experiments). Notice that we only need the feature vectors $\overline{\phi}(\mathbf{X})$ in EMK, hence it is not necessary to compute the entire kernel matrix. Since recent developments have shown that linear SVMs can be trained in linear complexity [25], there is no substantial cost added in the training phase. The complexity of EMK and of several other well-known set kernels is reviewed in table 1. If necessary, location information can be incorporated into EMK, using a spatial pyramid [14, 13]: $K_P(\mathbf{X}, \mathbf{Y}) = \sum_{l=0}^{L-1} \sum_{t=1}^{2^l} 2^{-l} K_S(\mathbf{X}^{(l,t)}, \mathbf{Y}^{(l,t)}) = \overline{\phi}_S(\mathbf{X})^\top \overline{\phi}_S(\mathbf{Y})$, where $L$ is the number of pyramid levels, $2^l$ is the number of spatial cells in the $l$-th pyramid level, $\mathbf{X}^{(l,s)}$ are local features falling within the spatial cell $(l, s)$, and $\overline{\phi}_P(\mathbf{X}) = [\overline{\phi}(\mathbf{X}^{(1,1)})^\top, \dots, \overline{\phi}(\mathbf{X}^{(l,s)})^\top]^\top$.

While there can be many choices for the local feature maps $\phi(\mathbf{x})$—and the positive definiteness of $k(\mathbf{x}, \mathbf{y}) = \phi(\mathbf{x})^\top \phi(\mathbf{x})$ can be always guaranteed—, most do not necessarily lead to a meaningful similarity measure. In the paper, we give two principled methods to create meaningful local feature maps $\phi(\mathbf{x})$, by arranging for their inner products to approximate a given kernel function.

## 3 Efficient Match Kernels

In this section we present two kernel approximations, based on low-dimensional projections (§3.1), and based on random Fourier set features (§3.2).

### 3.1 Learning Low Dimensional Set Features

Our approach is to project the high dimensional feature vectors $\psi(\mathbf{x})$ induced by the kernel $k(\mathbf{x}, \mathbf{y}) = \psi(\mathbf{x})^\top \psi(\mathbf{y})$ to a low dimensional space spanned by $D$ basis vectors, then construct a local kernel from inner products, based on low-dimensional representations. Given $\{\psi(\mathbf{z}_i)\}_{i=1}^D$, a set of basis vectors $\mathbf{z}_i$, we can approximate the feature vector $\psi(\mathbf{x})$:

$$\overline{\mathbf{v}}_\mathbf{x} = \underset{\mathbf{v}_\mathbf{x}}{\operatorname{argmin}} \|\psi(\mathbf{x}) - \mathbf{H}\mathbf{v}_\mathbf{x}\|^2 \tag{5}$$

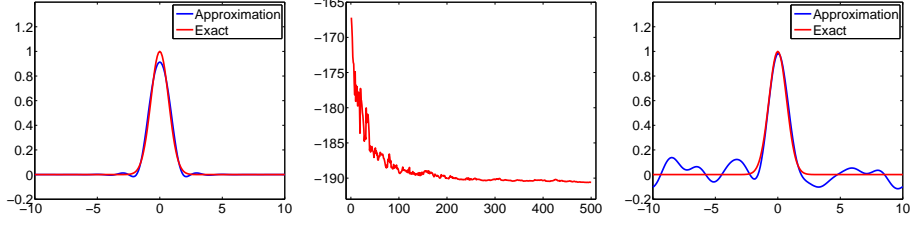

Figure 1: Low-dimensional approximations for a Gaussian kernel. *Left*: approximated Gaussian kernel with 20 learned feature maps. *Center*: the training objective (12) as a function of stochastic gradient descent iterations. *Right*: approximated Gaussian kernel based on 200 random Fourier features. The feature maps are learned from 200 samples, uniformly drawn from [-10,10].

where $\mathbf{H} = [\psi(\mathbf{z}_1), \ldots, \psi(\mathbf{z}_D)]$ and $\overline{\mathbf{v}}_{\mathbf{x}}$ are low-dimensional (projection) coefficients. This is a convex quadratic program with analytic solution:

$$\overline{\mathbf{v}}_{\mathbf{x}} = (\mathbf{H}^\top \mathbf{H})^{-1}(\mathbf{H}^\top \psi(\mathbf{x})) \tag{6}$$

The local kernel derived from the projected vectors is:

$$k_l(\mathbf{x}, \mathbf{y}) = [\mathbf{H}\overline{\mathbf{v}}_{\mathbf{x}}]^\top [\mathbf{H}\overline{\mathbf{v}}_{\mathbf{y}}] = \mathbf{k}_Z(\mathbf{x})^\top \mathbf{K}_{ZZ}^{-1} \mathbf{k}_Z(\mathbf{y}) \tag{7}$$

where $\mathbf{k}_Z$ is a $D \times 1$ vector with $\{\mathbf{k}_Z\}_i = k(\mathbf{x}, \mathbf{z}_i)$ and $\mathbf{K}_{ZZ}$ is a $D \times D$ matrix with $\{\mathbf{K}_{ZZ}\}_{ij} = k(\mathbf{z}_i, \mathbf{z}_j)$. For $\mathbf{G}^\top \mathbf{G} = \mathbf{K}_{ZZ}^{-1}$ (notice that $\mathbf{K}_{ZZ}^{-1}$ is positive definite), the local feature maps are:

$$\phi(\mathbf{x}) = \mathbf{G}\mathbf{k}_Z(\mathbf{x}) \tag{8}$$

The resulting full feature map is: $\overline{\phi}(\mathbf{X}) = \frac{1}{|\mathbf{X}|}\mathbf{G}\left[\sum_{\mathbf{x}\in\mathbf{X}} \mathbf{k}_Z(\mathbf{x})\right]$, with computational complexity $O(mDd + D^2)$ for a set of local features. A related method is the kernel codebook [28], where a set-level feature is also extracted based on a local kernel, but with different feature map $\overline{\phi}(\cdot)$. An essential difference is that inner products of our set-level features $\overline{\phi}(\mathbf{X})$ formally approximate the sum-match kernel, whereas the ones induced by the kernel codebook do not. Therefore EMK only requires a linear classifier whereas a kernel codebook would require a non-linear classifier for comparable performance. As explained, this can be prohibitively expensive to both train and test, in large datasets. Our experiments, shown in table 3, further suggest that EMK outperforms the kernel codebook, even in the non-linear case.

How can we learn the basis vectors? One way is kernel principal component analysis (KPCA) [24] on a randomly selected pool of $F$ local features, with the basis set to the topmost $D$ eigenvectors. This faces two difficulties, however: (*i*) KPCA scales cubically in the number of selected local features, $F$; (*ii*) $O(Fmd)$ work is required to extract the set-level feature vector for one image, because the eigenvectors are linear combinations of the selected local feature vectors, $\sum_{i=1}^{F} \alpha_i \psi(\mathbf{x}_i)$. For large $F$, as typically required for good accuracy, this approach is too expensive. Although the first difficulty can be palliated by iterative KPCA [11], the second computational challenge remains. Another option would be to approximate each eigenvector with a single feature vector $\psi(\overline{\mathbf{z}})$ by solving the pre-image problem $(\overline{\mathbf{z}}, \overline{\beta}) = \operatorname{argmin}_{\mathbf{z},\beta} \|\sum_{i=1}^{F} \alpha_i \psi(\mathbf{x}_i) - \beta\psi(\mathbf{z})\|^2$, after KPCA. However, the two step approach is sub-optimal. Intuitively, it should be better to find the single vector approximations within an unified objective function. This motivates our constrained singular value decomposition in kernel feature space (CKSVD):

$$\underset{\mathbf{V},\mathbf{Z}}{\operatorname{argmin}} R(\mathbf{V}, \mathbf{Z}) = \frac{1}{F}\sum_{i=1}^{F} \|\psi(\mathbf{x}_i) - \mathbf{H}\mathbf{v}_i\|^2 \tag{9}$$

where $F$ is the number of the randomly selected local features, $\mathbf{Z} = [\mathbf{z}_1, \ldots, \mathbf{z}_D]$ and $\mathbf{V} = [\mathbf{v}_1, \ldots, \mathbf{v}_F]$. If the pre-image constraints $\mathbf{H} = [\psi(\mathbf{z}_1), \ldots, \psi(\mathbf{z}_D)]$ are dropped, it is easy to show that KPCA can be recovered. The partial derivatives of $R$ with respect to $\mathbf{v}_i$ are:

$$\frac{\partial R(\mathbf{V}, \mathbf{Z})}{\partial \mathbf{v}_i} = 2\mathbf{H}^\top \mathbf{H}\mathbf{v}_i - 2\mathbf{H}^\top \psi(\mathbf{x}_i) \tag{10}$$

Expanding equalities like $\frac{\partial R(\mathbf{V}, \mathbf{Z})}{\partial \mathbf{v}_i} = 0$ produces a linear system with respect to $\mathbf{v}_i$ for a fixed $\mathbf{Z}$. In this case, we can obtain the optimal, analytical solution: $\overline{\mathbf{v}}_i = (\mathbf{H}^\top \mathbf{H})^{-1}(\mathbf{H}^\top \psi(\mathbf{x}_i))$. Substituting the solution in eq. (9), we can eliminate the variable $\mathbf{V}$. To learn the basis vectors, instead of directly optimizing $R(\mathbf{V}, \mathbf{Z})$, we can solve the equivalent optimization problem:

$$\underset{\mathbf{Z}}{\operatorname{argmin}} \, R^*(\mathbf{Z}) = -\frac{1}{F} \sum_{i=1}^{F} \mathbf{k}_Z(\mathbf{x}_i)^\top \mathbf{K}_{ZZ}^{-1} \mathbf{k}_Z(\mathbf{x}_i) \tag{11}$$

Optimizing $R^*(\mathbf{Z})$ is tractable because its parameter space is much smaller than $R(\mathbf{V}, \mathbf{Z})$. The problem (11) can be solved using any gradient descent algorithm. For efficiency, we use the stochastic (on-line) gradient descent (SGD) method. SGD applies to problems where the full gradient decomposes as a sum of individual gradients of the training samples. The standard (batch) gradient descent method updates the parameter vector using the full gradient whereas SGD approximates it using the gradient at a single training sample. For large datasets, SGD is usually much faster than batch gradient descent. At the $t$-th iteration, in SCG, we randomly pick a sample $\mathbf{x}_t$ from the training set and update the parameter vector based on:

$$\mathbf{Z(t+1)} = \mathbf{Z(t)} - \frac{\eta}{t} \frac{\partial \left[ -\mathbf{k}_Z(\mathbf{x}_t)^\top \mathbf{K}_{ZZ}^{-1} \mathbf{k}_Z(\mathbf{x}_t) \right]}{\partial \mathbf{Z}} \tag{12}$$

where $\eta$ is the learning rate. In our implementation, we use $D$ samples (rather than just one) to compute the gradient. This produces more accurate results and matches the cost of inverting $\mathbf{K}_{ZZ}$, which is $O(D^3)$ per iteration.

## 3.2 Random Fourier Set Features

Another tractable approach to large-scale learning is to approximate the kernel using random feature maps [22, 23]. For a given function $\mu(\mathbf{x}; \theta)$ and the probability distribution $p(\theta)$, one can define the local kernel as: $k_f(\mathbf{x}, \mathbf{y}) = \int p(\theta) \mu(\mathbf{x}; \theta) \mu(\mathbf{y}, \theta) d\theta$. We consider feature maps of the form $\mu(\mathbf{x}; \theta) = \cos(\omega^\top \mathbf{x} + b)$ with $\theta = (\omega, b)$, which project local features to a randomly chosen line, then pass the resulting scalar through a sinusoid. For example, to approximate the Gaussian kernel $k_f(\mathbf{x}, \mathbf{y}) = \exp(-\gamma \|\mathbf{x} - \mathbf{y}\|^2)$, the random feature maps are: $\phi(\mathbf{x}) = \sqrt{\frac{2}{D}}[\cos(\omega_1^\top \mathbf{x} + b_1), \ldots, \cos(\omega_D^\top \mathbf{x} + b_D)]^\top$, where $b_i$ are drawn from the uniform distribution $[-\pi, \pi]$ and $\omega$ are drawn from a Gaussian with 0 mean and covariance $2\gamma \mathbf{I}$. Our proposed set-level feature map is (*c.f.* §2): $\overline{\phi}(\mathbf{X}) = \frac{1}{|\mathbf{X}|} \sum_{\mathbf{x} \in \mathbf{X}} \phi(\mathbf{x})$. Although any shift invariant kernel can be represented using random Fourier features, currently these are limited to Gaussian kernels or to kernels with analytical inverse Fourier transforms. In particular, $\omega$ needs to be sampled from the inverse Fourier transform of the corresponding shift invariant kernel. The constraint of a shift-invariant kernel excludes a number of practically interesting similarities. For example, the $\chi^2$ kernel [8] and the histogram intersection kernel [5] are designed to compare histograms, hence they can be used as local kernels, if the features are histograms. However, no random Fourier features can approximate them. Such problems do not occur for the learned low dimensional features—a methodology applicable to any Mercer kernel. Moreover, in experiments, we show that kernels based on low-dimensional approximations (§3.1) can produce superior results when the dimensionality of the feature maps is small. As seen in fig. 2, for applicable kernels, the random Fourier set features also produce very competitive results in the higher-dimensional regime.

## 4 Experiments

We illustrate our methodology in three publicly available computer vision datasets: Scene-15, Caltech-101 and Caltech-256. For comparisons, we consider four algorithms: BOW-Linear, BOW-Gaussian, EMK-CKSVD and EMK-Fourier. BOW-Linear and BOW-Gaussian use a linear classifier and a Gaussian kernel classifier on BOW features, respectively. EMK-CKSVD and EMK-Fourier use linear classifiers. For the former, we learn low dimensional feature maps (§3.1), whereas for the latter we obtain them using random sampling (§3.2).

All images are transformed into grayscale form. The local features are SIFT descriptors [16] extracted from 16×16 image patches. Instead of detecting the interest points, we compute SIFT descriptors over dense regular grids with spacing of 8 pixels. For EMK, our local kernel is a Gaussian

$\exp(-\gamma\|\mathbf{x} - \mathbf{y}\|^2)$. We use the same fixed $\gamma = 1$ for our SIFT descriptor in all datasets: Scene-15, Caltech-101 and Caltech-256, although a more careful selection is likely to further improve performance. We run $k$-means clustering to identify the visual words and stochastic gradient descent to learn the local feature maps, using a 100,000 random set of SIFT descriptors.

Our classifier is a support vector machine (SVM), which is extended to multi-class decisions by combining one-versus-all votes. We work with LIBLINEAR [3] for BOW-Linear, EMK-Fourier and EMK-CKSVD, and LIBSVM for BOW-Gaussian (the former need a linear classifier whereas the latter uses a nonlinear classifier). The regularization and the kernel parameters (if available) in SVM are tuned by ten-fold cross validation on the training set. The dimensionality of the feature maps and the vocabulary size are both set to 1000 for fair comparisons, unless otherwise specified. We have also experimented with larger vocabulary sizes in BOW, but no substantial improvement was found (fig. 2). We measure performance based on classification accuracy, averaged over five random training/testing splits. All experiments are run on a cluster built of compute nodes with 1.0 GHz processors and 8GB memory.

**Scene-15:** Scene-15 consists of 4485 images labeled into 15 categories. Each category contains 200 to 400 images whose average size is 300×250 pixels. In our first experiment, we train models on a randomly selected set of 1500 images (100 images per category) and test on the remaining images. We vary the dimensionality of the feature maps (EMK) and the vocabulary size (BOW) from 250 to 2000 with step length 250. For this dataset, we only consider the flat BOW and EMK (only pyramid level 0) in all experiments. The classification accuracy of BOW-Linear, BOW-Gaussian, EMK-Fourier and EMK-CKSVD is plotted in fig. 2 (*left*). Our second experiment is similar with the first one, but the dimensionality of the feature maps and the vocabulary size vary from 50 to 200 with step length 25. In our third experiment, we fix the dimensionality of the feature maps to 1000, and vary the training set size from 300 to 2400 with step length 300. We show the classification accuracy of the four models as a function of the training set size in fig. 2 (*right*).

We notice that EMK is consistently 5-8% better than BOW in all cases. BOW-Gaussian is about 2 % better than BOW-Linear on average, whereas EMK-CKSVD give are very similar performance to EMK-Fourier in most cases. We observe that EMK-CKSVD significantly outperforms EMK-Fourier for low-dimensional feature maps, indicating that learned features preserve the values of the Gaussian kernel better than the random Fourier maps in this regime, see also fig. 1 (*center*).

For comparisons, we attempted to run the sum match kernel, on the full Scene-15 dataset. However, we weren't able to finish in one week. Therefore, we considered a smaller dataset, by training and testing with only 40 images from each category. The sum match kernel obtains $71.8\%$ accuracy and slightly better than EMK-Fourier $71.0\%$ and EMK-CKSVD $71.4\%$ on the same dataset. The sum match kernel takes about 10 hours for training and 10 hours for testing, respectively whereas EMK-Fourier and EMK-CKSVD need less than 1 hour, most spent computing SIFT descriptors. In addition, we use 10,000 randomly selected SIFT descriptors to learn KPCA-based local feature maps, which takes about 12 hours for the training and testing sets on the full Scene-15 dataset, respectively. We obtain slightly lower accuracy than EMK-Fourier and EMK-CKSVD. One reason can be the small sample size, but it is currently prohibitive, computationally, to use larger ones.

**Caltech-101:** Caltech-101 [15] contains 9144 images from 101 object categories and a background category. Each category has 31 to 800 images with significant color, pose and lighting variations. Caltech-101 is one of the most frequently used benchmarks for image classification, and results obtained by different algorithms are available from the published papers, allowing direct comparisons. Following the common experimental setting, we train models on 15/30 image per category and test on the remaining images. We consider three pyramid levels: $L = 0$, $L = 1$, amd $L = 2$ (for the latter two, spatial information is used). We have also tried increasing the number of levels in the pyramid, but did not obtain a significant improvement.

We report the accuracy of BOW-Linear, BOW-Gaussian, EMK-Fourier and EMK-CKSVD in table 2. EMK-Fourier and EMK-CKSVD perform substantially better than BOW-Linear and BOW-Gaussian for all pyramid levels. The performance gap increases as more pyramid levels are added. EMK-CKSVD is very close to EMK-Fourier and BOW-Gaussian does not improve over BOW-Linear much. In table 3, we compare EMK to other algorithms. As we have seen, EMK is comparable to the best-scoring classifiers to date. The best result on Caltech101 was obtained by combining multiple descriptor types [1]. Our main goal in this paper is to analyze the strengths of EMK relative

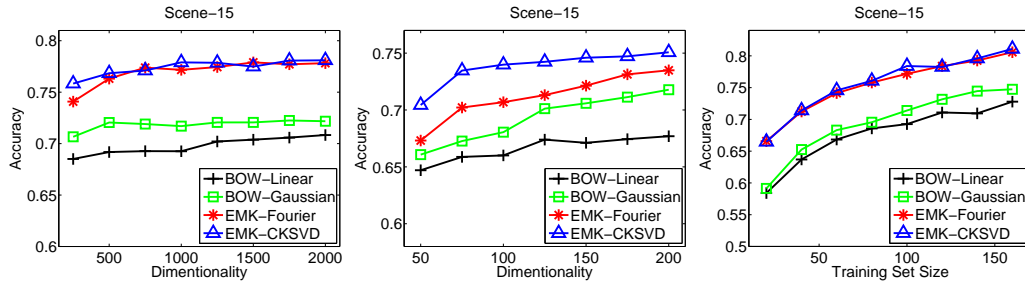

Figure 2: Classification accuracy on Scene-15. *Left*: Accuracy in the high-dimensional regime, and (*center*) in the low-dimensional regime. *Right*: Accuracy as a function of the training set size. The training set size is 1500 in the left plot; the dimensionality of feature maps and the vocabulary size are both set to 1000 in the right plot (for fair comparisons).

| Algorithms | Pyramid levels(15 training) | | | Pyramid levels (30 training) | | |
|---|---|---|---|---|---|---|
| | L=0 | L=1 | L=2 | L=0 | L=1 | L=2 |
| BOW-Linear | 37.3± 0.9 | 41.6± 0.7 | 45.0±0.5 | 46.2±0.8 | 53.0± 0.9 | 56.2±0.7 |
| BOW-Gaussian | 38.7± 0.8 | 43.7± 0.7 | 46.5±0.6 | 47.5±0.7 | 54.7± 0.8 | 58.1±0.6 |
| EMK-Fourier | 46.3± 0.7 | 53.0± 0.6 | 60.2±0.8 | 54.0± 0.7 | 64.1± 0.8 | 70.1±0.8 |
| EMK-CKSVD | 46.6±0.9 | 53.4±0.8 | 60.5±0.9 | 54.5±0.8 | 63.7±0.9 | 70.3±0.8 |

Table 2: Classification accuracy comparisons for three pyramid levels. The results are averaged over five random training/testing splits. The dimensionality of the feature maps and the vocabulary size are both set to 1000. We have also experimented with large vocabularies, but did not observe noticeable improvement—the performance tends to saturate beyond 1000 dimensions.

to BOW. Only SIFT descriptors are used in BOW and EMK for all compared algorithms, listed in table 3. To improve performance, EMK can be conveniently extended to multiple feature types.

**Caltech-256:** Caltech-256 consists of 30,607 images from 256 object categories and background, where each category contains at least 80 images. Caltech-256 is challenging due to the large number of classes and the diverse lighting conditions, poses, backgrounds, images size, *etc*. We follow the standard setup and increase the training set from 15 to 60 images per category with step length 15. In table 4, we show the classification accuracy obtained from BOW-Linear, BOW-Gaussian, EMK-Fourier and EMK-CKSVD. As in the other datasets, we notice that EMK-Fourier and EMK-CKSVD consistently outperform the BOW-Linear and the BOW-Gaussian.

To compare the four algorithms computationally, we select images from each category proportionally to the total number of images of that category, as the training set. We consider six different training set sizes: $\lfloor 0.3 \times 30607 \rfloor, \ldots, \lfloor 0.8 \times 30607 \rfloor$. The results are shown in fig. 3. To accelerate BOW-Gaussian, we precompute the entire kernel matrix. As expected, BOW-Gaussian is much slower than the other three algorithms as the training set size increases, for both training and testing.

| Algorithms | 15 training | 30 training | Algorithms | 15 training | 30 training |
|---|---|---|---|---|---|
| PMK [5, 6] | 50.0±0.9 | 58.2 | kCNN [30] | 59.2 | 67.4 |
| HMAX [19] | 51.0 | 56.0 | LDF [4] | 60.3 | N/A |
| ML+PMK [9] | 52.2 | 62.1 | ML+CORR [9] | 61.0 | 69.6 |
| KC [28] | N/A | 64.0 | NBNN [1] | 65.0±1.1 | 73.0 |
| SPM [14] | 56.4 | 64.4±0.5 | EMK-Fourier | 60.2±0.8 | 70.1±0.8 |
| SVM-KNN [31] | 59.1±0.5 | 66.2±0.8 | EMK-CKSVD | 60.5±0.9 | 70.3±0.8 |

Table 3: Accuracy comparisons on Caltech-101. EMK is compared with ten recently published methods. N/A indicates that results are not available. Notice that EMK is used in conjunction with a linear classifier (linear SVM here) whereas all other methods (except HMAX [19]) require nonlinear classifiers.

| Algorithms | BOW-Linear | BOW-Gaussian | EMK-Fourier | EMK-CKSVD |
|---|---|---|---|---|
| 15 training | 17.4±0.7 | 19.1±0.8 | 22.6±0.7 | 23.2±0.6 |
| 30 training | 22.7±0.4 | 24.4±0.6 | 30.1±0.5 | 30.5±0.4 |
| 45 training | 26.9±0.3 | 28.3±0.5 | 34.1±0.5 | 34.4±0.4 |
| 60 training | 29.3±0.6 | 30.9±0.4 | 37.4±0.6 | 37.6±0.5 |

Table 4: Accuracy on Caltech-256. The results are averaged over five random training/testing splits. The dimensionality of the feature maps and the vocabulary size are both set to 1000 (for fair comparisons). We use 2 pyramid levels.

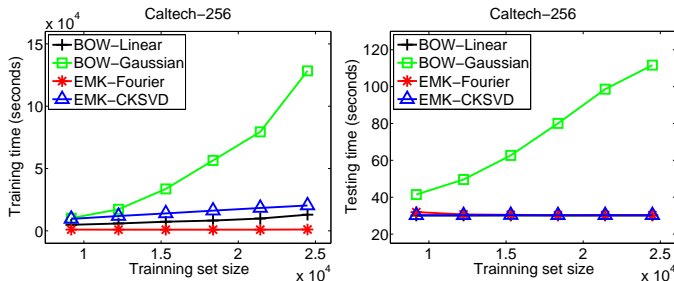

Figure 3: Computational costs on Caltech-256. *Left*: training time as a function of the training set size. *Right*: testing time as a function of the training set size. Testing time is in seconds per 100 samples. Flat BOW and EMK are used (no pyramid, $L = 0$). Notice that PMK has a similar training and testing cost with BOW-Gaussian.

Nonlinear SVMs takes $O(n^2 \sim n^3)$ even when a highly optimized software package like LIBSVM is used. For large $n$, the SVM training dominates the training cost. The testing time of BOW-Gaussian is linear in the training set size, but constant for the other three algorithms. Although we only experiment with a Gaussian kernel, a similar complexity would be typical for other nonlinear kernels, as used in [6, 9, 14, 31, 4].

## 5   Conclusion

We have presented efficient match kernels for visual recognition, based on a novel insight that popular bag-of-words representations used in conjunction with linear models can be viewed as a special type of match kernel which counts 1 if two local features fall into the same regions partitioned by visual words and 0 otherwise. We illustrate the quantization limitations of such models and propose more sophisticated kernel approximations that preserve the computational efficiency of bag-of-words while being just as (or more) accurate than the existing, computationally demanding, non-linear kernels. The models we propose are built around Efficient Match Kernels (EMK), which map local features to a low dimensional feature space, average the resulting feature vectors to form a set-level feature, then apply a linear classifier. In experiments, we show that EMK are efficient and achieve state of the art classification results in three difficult computer vision datasets: Scene-15, Caltech-101 and Caltech-256.

**Acknowledgements:** This research was supported, in part, by awards from NSF (IIS-0535140) and the European Commission (MCEXT-025481). Liefeng Bo thanks Jian Peng for helpful discussions.

## References

[1] O. Boiman, E. Shechtman, and M. Irani. In defense of nearest-neighbor based image classification. In *CVPR*, 2008.

[2] M. Cuturi and J. Vert. Semigroup kernels on finite sets. In *NIPS*, 2004.

[3] R. Fan, K. Chang, C. Hsieh, X. Wang, and C. Lin. Liblinear: A library for large linear classification. *JMLR*, 9:1871–1874, 2008.

[4] A. Frome, Y. Singer, and J. Malik. Image retrieval and classification using local distance functions. In *NIPS*, 2006.

[5] K. Grauman and T. Darrell. The pyramid match kernel: discriminative classification with sets of image features. In *ICCV*, 2005.

[6] K. Grauman and T. Darrell. The pyramid match kernel: Efficient learning with sets of features. *JMLR*, 8:725–760, 2007.

[7] D. Haussler. Convolution kernels on discrete structures. Technical report, 1999.

[8] Zhang J., Marszalek M., Lazebnik S., and Schmid C. Local features and kernels for classification of texture and object categories: A comprehensive study. *IJCV*, 73(2):213–238, 2007.

[9] P. Jain, B. Kulis, and K. Grauman. Fast image search for learned metrics. In *CVPR*, 2008.

[10] F. Jurie and B. Triggs. Creating efficient codebooks for visual recognition. In *ICCV*, 2005.

[11] K. Kim, M. Franz, and B. Schölkopf. Iterative kernel principal component analysis for image modeling. *PAMI*, 27(9):1351–1366, 2005.

[12] R. Kondor and T. Jebara. A kernel between sets of vectors. In *ICML*, 2003.

[13] A. Kumar and C. Sminchisescu. Support kernel machines for object recognition. In *ICCV*, 2007.

[14] S. Lazebnik, C. Schmid, and J. Ponce. Beyond bags of features: Spatial pyramid matching for recognizing natural scene categories. In *CVPR*, 2006.

[15] F. Li, R. Fergus, and P. Perona. One-shot learning of object categories. *PAMI*, 28(4):594–611, 2006.

[16] D. Lowe. Distinctive image features from scale-invariant keypoints. *IJCV*, 60:91–110, 2004.

[17] S. Lyu. Mercer kernels for object recognition with local features. In *CVPR*, 2005.

[18] P. Moreno, P. Ho, and N. Vasconcelos. A kullback-leibler divergence based kernel for svm classification in multimedia applications. In *NIPS*, 2003.

[19] J. Mutch and D. Lowe. Multiclass object recognition with sparse, localized features. In *CVPR*, 2006.

[20] M. Parsana, S. Bhattacharya, C. Bhattacharyya, and K. Ramakrishnan. Kernels on attributed pointsets with applications. In *NIPS*, 2007.

[21] J. Philbin, O. Chum, M. Isard, J. Sivic, and A. Zisserman. Lost in quantization: Improving particular object retrieval in large scale image databases. In *CVPR*, 2008.

[22] A. Rahimi and B. Recht. Random features for large-scale kernel machines. In *NIPS*, 2007.

[23] A. Rahimi and B. Recht. Weighted sums of random kitchen sinks: Replacing minimization with randomization in learning. In *NIPS*, 2008.

[24] B. Schölkopf, A. Smola, and K. Müller. Nonlinear component analysis as a kernel eigenvalue problem. *Neural Computation*, 10:1299–1319, 1998.

[25] S. Shalev-Shwartz, Y. Singer, and N. Srebro. Pegasos: Primal estimated sub-gradient solver for svm. In *ICML*, pages 807–814. ACM, 2007.

[26] A. Shashua and T. Hazan. Algebraic set kernels with application to inference over local image representations. In *NIPS*, 2004.

[27] J. Sivic and A. Zisserman. Video google: A text retrieval approach to object matching in videos. In *ICCV*, 2003.

[28] J. van Gemert, J. Geusebroek, C. Veenman, and A. Smeulders. Kernel codebooks for scene categorization. In *ECCV*, 2008.

[29] L. Wolf and A. Shashua. Learning over sets using kernel principal angles. *JMLR*, 4:913–931, 2003.

[30] K. Yu, W. Xu, and Y. Gong. Deep learning with kernel regularization for visual recognition. In *NIPS*, 2008.

[31] H. Zhang, A. Berg, M. Maire, and J. Malik. Svm-knn: Discriminative nearest neighbor classification for visual category recognition. In *CVPR*, 2006.

